# Bayes-Adaptive POMDPs

**Stéphane Ross**
McGill University
Montréal, Qc, Canada
sross12@cs.mcgill.ca

**Brahim Chaib-draa**
Laval University
Québec, Qc, Canada
chaib@ift.ulaval.ca

**Joelle Pineau**
McGill University
Montréal, Qc, Canada
jpineau@cs.mcgill.ca

## Abstract

Bayesian Reinforcement Learning has generated substantial interest recently, as it provides an elegant solution to the exploration-exploitation trade-off in reinforcement learning. However most investigations of Bayesian reinforcement learning to date focus on the standard Markov Decision Processes (MDPs). Our goal is to extend these ideas to the more general Partially Observable MDP (POMDP) framework, where the state is a hidden variable. To address this problem, we introduce a new mathematical model, the Bayes-Adaptive POMDP. This new model allows us to (1) improve knowledge of the POMDP domain through interaction with the environment, and (2) plan optimal sequences of actions which can trade-off between improving the model, identifying the state, and gathering reward. We show how the model can be finitely approximated while preserving the value function. We describe approximations for belief tracking and planning in this model. Empirical results on two domains show that the model estimate and agent's return improve over time, as the agent learns better model estimates.

## 1 Introduction

In many real world systems, uncertainty can arise in both the prediction of the system's behavior, and the observability of the system's state. Partially Observable Markov Decision Processes (POMDPs) take both kinds of uncertainty into account and provide a powerful model for sequential decision making under these conditions. However most solving methods for POMDPs assume that the model is known a priori, which is rarely the case in practice. For instance in robotics, the POMDP must reflect exactly the uncertainty on the robot's sensors and actuators. These parameters are rarely known exactly and therefore must often be approximated by a human designer, such that even if this approximate POMDP could be solved exactly, the resulting policy may not be optimal. Thus we seek a decision-theoretic planner which can take into account the uncertainty over model parameters during the planning process, as well as being able to learn from experience the values of these unknown parameters.

Bayesian Reinforcement Learning has investigated this problem in the context of fully observable MDPs [1, 2, 3]. An extension to POMDP has recently been proposed [4], yet this method relies on heuristics to select actions that will improve the model, thus forgoing any theoretical guarantee on the quality of the approximation, and on an oracle that can be queried to provide the current state.

In this paper, we draw inspiration from the Bayes-Adaptive MDP framework [2], which is formulated to provide an optimal solution to the exploration-exploitation trade-off. To extend these ideas to POMDPs, we face two challenges: (1) how to update Dirichlet parameters when the state is a hidden variable? (2) how to approximate the infinite dimensional belief space to perform belief monitoring and compute the optimal policy. This paper tackles both problem jointly. The first problem is solved by including the Dirichlet parameters in the state space and maintaining belief states over these parameters. We address the second by bounding the space of Dirichlet parameters to a finite subspace necessary for $\epsilon$-optimal solutions.

We provide theoretical results for bounding the state space while preserving the value function and we use these results to derive approximate solving and belief monitoring algorithms. We compare several belief approximations in two problem domains. Empirical results show that the agent is able to learn good POMDP models and improve its return as it learns better model estimate.

## 2  POMDP

A POMDP is defined by finite sets of states $S$, actions $A$ and observations $Z$. It has transition probabilities $\{T^{sas'}\}_{s,s'\in S, a\in A}$ where $T^{sas'} = \Pr(s_{t+1} = s'|s_t = s, a_t = a)$ and observation probabilities $\{O^{saz}\}_{s\in S, a\in A, z\in Z}$ where $O^{saz} = \Pr(z_t = z|s_t = s, a_{t-1} = a)$. The reward function $R : S \times A \to \mathbb{R}$ specifies the immediate reward obtained by the agent. In a POMDP, the state is never observed. Instead the agent perceives an observation $z \in Z$ at each time step, which (along with the action sequence) allows it to maintain a belief state $b \in \Delta S$. The belief state specifies the probability of being in each state given the history of action and observation experienced so far, starting from an initial belief $b_0$. It can be updated at each time step using Baye's rule: $b_{t+1}(s') =$

$$\frac{O^{s'a_tz_{t+1}} \sum_{s\in S} T^{sa_ts'} b_t(s)}{\sum_{s''\in s} O^{s''a_tz_{t+1}} \sum_{s\in S} T^{sa_ts''} b_t(s)}.$$

A policy $\pi : \Delta S \to A$ indicates how the agent should select actions as a function of the current belief. Solving a POMDP involves finding the optimal policy $\pi^*$ that maximizes the expected discounted return over the infinite horizon. The return obtained by following $\pi^*$ from a belief $b$ is defined by Bellman's equation: $V^*(b) = \max_{a\in A} \left[ \sum_{s\in S} b(s)R(s,a) + \gamma \sum_{z\in Z} \Pr(z|b,a) V^*(\tau(b,a,z)) \right]$, where $\tau(b,a,z)$ is the new belief after performing action $a$ and observation $z$ and $\gamma \in [0,1)$ is the discount factor.

Exact solving algorithms [5] are usually intractable, except on small domains with only a few states, actions and observations. Various approximate algorithms, both offline [6, 7, 8] and online [9], have been proposed to tackle increasingly large domains. However, all these methods requires full knowledge of the POMDP model, which is a strong assumption in practice. Some approaches do not require knowledge of the model, as in [10], but these approaches generally require a lot of data and do not address the exploration-exploitation tradeoff.

## 3  Bayes-Adaptive POMDP

In this section, we introduce the Bayes-Adaptive POMDP (BAPOMDP) model, an optimal decision-theoretic algorithm for learning and planning in POMDPs under parameter uncertainty. Throughout we assume that the state, action, and observation spaces are finite and known, but that the transition and observation probabilities are unknown or partially known. We also assume that the reward function is known as it is generally specified by the user for the specific task he wants to accomplish, but the model can easily be generalised to learn the reward function as well.

To model the uncertainty on the transition $T^{sas'}$ and observation $O^{saz}$ parameters, we use *Dirichlet distributions*, which are probability distributions over the parameters of multinomial distributions. Given $\phi_i$, the number of times event $e_i$ has occurred over $n$ trials, the probabilities $p_i$ of each event follow a Dirichlet distribution, i.e. $(p_1, \ldots, p_k) \sim Dir(\phi_1, \ldots, \phi_k)$. This distribution represents the probability that a discrete random variable behaves according to some probability distribution $(p_1, \ldots, p_k)$, given that the counts $(\phi_1, \ldots, \phi_k)$ have been observed over $n$ trials ($n = \sum_{i=1}^{k} \phi_i$). Its probability density function is defined by: $f(p, \phi) = \frac{1}{B(\phi)} \prod_{i=1}^{k} p_i^{\phi_i - 1}$, where $B$ is the multinomial beta function. The expected value of $p_i$ is $\mathrm{E}(p_i) = \frac{\phi_i}{\sum_{j=1}^{k} \phi_j}$.

### 3.1  The BAPOMDP Model

The BAPOMDP is constructed from the model of the POMDP with unknown parameters. Let $(S, A, Z, T, O, R, \gamma)$ be that model. The uncertainty on the distributions $T^{sa\cdot}$ and $O^{s'a\cdot}$ can be represented by experience counts: $\phi_{ss'}^a, \forall s'$ represents the number of times the transition $(s, a, s')$ occurred, similarly $\psi_{s'z}^a, \forall z$ is the number of times observation $z$ was made in state $s'$ after doing action $a$. Let $\phi$ be the vector of all transition counts and $\psi$ be the vector of all observation counts. Given

the count vectors $\phi$ and $\psi$, the expected transition probability for $T^{sas'}$ is: $T_\phi^{sas'} = \frac{\phi_{ss'}^a}{\sum_{s'' \in S} \phi_{ss''}^a}$, and similarly for $O^{s'az}$: $O_\psi^{s'az} = \frac{\psi_{s'z}^a}{\sum_{z' \in Z} \psi_{s'z'}^a}$.

The objective of the BAPOMDP is to learn an optimal policy, such that actions are chosen to maximize reward taking into account both state and parameter uncertainty. To model this, we follow the Bayes-Adaptive MDP framework, and include the $\phi$ and $\psi$ vectors in the state of the BAPOMDP. Thus, the state space $S'$ of the BAPOMDP is defined as $S' = S \times \mathcal{T} \times \mathcal{O}$, where $\mathcal{T} = \{\phi \in \mathbb{N}^{|S|^2|A|} | \forall(s,a), \sum_{s' \in S} \phi_{ss'}^a > 0\}$ represents the space in which $\phi$ lies and $\mathcal{O} = \{\psi \in \mathbb{N}^{|S||A||Z|} | \forall(s,a), \sum_{z \in Z} \psi_{sz}^a > 0\}$ represents the space in which $\psi$ lies. The action and observation sets of the BAPOMDP are the same as in the original POMDP. Transition and observation functions of the BAPOMDP must capture how the state and count vectors $\phi$, $\psi$ evolve after every time step. Consider an agent in a given state $s$ with count vectors $\phi$ and $\psi$, which performs action $a$, causing it to move to state $s'$ and observe $z$. Then the vector $\phi'$ after the transition is defined as $\phi' = \phi + \delta_{ss'}^a$, where $\delta_{ss'}^a$ is a vector full of zeroes, with a 1 for the count $\phi_{ss'}^a$, and the vector $\psi'$ after the observation is defined as $\psi' = \psi + \delta_{s'z}^a$, where $\delta_{s'z}^a$ is a vector full of zeroes, with a 1 for the count $\psi_{s'z}^a$. Note that the probabilities of such transitions and observations occurring must be defined by considering all models and their probabilities as specified by the current Dirichlet distributions, which turn out to be their expectations. Hence, we define $T'$ and $O'$ to be:

$$T'((s,\phi,\psi),a,(s',\phi',\psi')) = \begin{cases} T_\phi^{sas'} O_\psi^{s'az}, & \text{if } \phi' = \phi + \delta_{ss'}^a \text{ and } \psi' = \psi + \delta_{s'z}^a \\ 0, & \text{otherwise.} \end{cases} \qquad (1)$$

$$O'((s,\phi,\psi),a,(s',\phi',\psi'),z) = \begin{cases} 1, & \text{if } \phi' = \phi + \delta_{ss'}^a \text{ and } \psi' = \psi + \delta_{s'z}^a \\ 0, & \text{otherwise.} \end{cases} \qquad (2)$$

Note here that the observation probabilities are folded into the transition function, and that the observation function becomes deterministic. This happens because a state transition in the BAPOMDP automatically specifies which observation is acquired after transition, via the way the counts are incremented. Since the counts do not affect the reward, the reward function of the BAPOMDP is defined as $R'((s,\phi,\psi),a) = R(s,a)$; the discount factor of the BAPOMDP remains the same. Using these definitions, the BAPOMDP has a known model specified by the tuple $(S', A, Z, T', O', R', \gamma)$.

The belief state of the BAPOMDP represents a distribution over both states and count values. The model is learned by simply maintaining this belief state, as the distribution will concentrate over most likely models, given the prior and experience so far. If $b_0$ is the initial belief state of the unknown POMDP, and the count vectors $\phi_0 \in \mathcal{T}$ and $\psi_0 \in \mathcal{O}$ represent the prior knowledge on this POMDP, then the initial belief of the BAPOMDP is: $b_0'(s, \phi_0, \psi_0) = \{b_0(s), \text{if } (\phi, \psi) = (\phi_0, \psi_0); 0, \text{otherwise}\}$. After actions are taken, the uncertainty on the POMDP model is represented by mixtures of Dirichlet distributions (i.e. mixtures of count vectors).

Note that the BAPOMDP is in fact a POMDP with a countably infinite state space. Hence the belief update function and optimal value function are still defined as in Section 2. However these functions now require summations over $S' = S \times \mathcal{T} \times \mathcal{O}$. Maintaining the belief state is practical only if the number of states with non-zero probabilities is finite. We prove this in the following theorem:

**Theorem 3.1.** *Let* $(S', A, Z, T', O', R', \gamma)$ *be a BAPOMDP constructed from the POMDP* $(S, A, Z, T, O, R, \gamma)$. *If $S$ is finite, then at any time $t$, the set* $S'_{b_t'} = \{\sigma \in S' | b_t'(\sigma) > 0\}$ *has size* $|S'_{b_t'}| \leq |S|^{t+1}$.

*Proof.* Proof available in [11]. Proceeds by induction from $b_0'$. $\qquad\qquad\square$

The proof of this theorem suggests that it is sufficient to iterate over $S$ and $S'_{b_{t-1}'}$ in order to compute the belief state $b_t'$ when an action and observation are taken in the environment. Hence, Algorithm 3.1 can be used to update the belief state.

## 3.2 Exact Solution for BAPOMDP in Finite Horizons

The value function of a BAPOMDP for finite horizons can be represented by a finite set $\Gamma$ of functions $\alpha : S' \rightarrow \mathbb{R}$, as in standard POMDP. For example, an exact solution can be computed using

```
function τ(b, a, z)
  Initialize b' as a 0 vector.
  for all (s, φ, ψ, s') ∈ S'_b × S do
      b'(s', φ + δ^a_{ss'}, ψ + δ^a_{s'z}) ← b'(s', φ + δ^a_{ss'}, ψ + δ^a_{s'z}) + b(s, φ, ψ)T^{sas'}_φ O^{s'az}_ψ
  end for
  return normalized b'
```

**Algorithm 3.1:** Exact Belief Update in BAPOMDP.

dynamic programming (see [5] for more details):

$$
\begin{aligned}
\Gamma^a_1 &= \{\alpha^a | \alpha^a(s, \phi, \psi) = R(s, a)\}, \\
\Gamma^{a,z}_t &= \{\alpha^{a,z}_i | \alpha^{a,z}_i(s, \phi, \psi) = \gamma \sum_{s' \in S} T^{sas'}_\phi O^{s'az}_\psi \alpha'_i(s', \phi + \delta^a_{ss'}, \psi + \delta^a_{s'z}), \alpha'_i \in \Gamma_{t-1}\}, \\
\Gamma^a_t &= \Gamma^a_1 \oplus \Gamma^{a,z_1}_t \oplus \Gamma^{a,z_2}_t \oplus \cdots \oplus \Gamma^{a,z_{|Z|}}_t, \quad \text{(where } \oplus \text{ is the cross sum operator)}, \\
\Gamma_t &= \bigcup_{a \in A} \Gamma^a_t.
\end{aligned}
\tag{3}
$$

Note here that the definition of $\alpha^{a,z}_i(s, \phi, \psi)$ is obtained from the fact that $T'((s, \phi, \psi), a, (s', \phi', \psi'))O'((s, \phi, \psi), a, (s', \phi', \psi'), z) = 0$ except when $\phi' = \phi + \delta^a_{ss'}$ and $\psi' = \psi + \delta^a_{s'z}$. The optimal policy is extracted as usual: $\pi_\Gamma(b) = \text{argmax}_{\alpha \in \Gamma} \sum_{\sigma \in S'_b} \alpha(\sigma)b(\sigma)$. In practice, it will be impossible to compute $\alpha^{a,z}_i(s, \phi, \psi)$ for all $(s, \phi, \psi) \in S'$. In order to compute these more efficiently, we show in the next section that the infinite state space can be reduced to a finite state space, while still preserving the value function to arbitrary precision for any horizon $t$.

# 4 Approximating the BAPOMDP: Theory and Algorithms

Solving a BAPOMDP exactly for all belief states is impossible in practice due to the dimensionnality of the state space (in particular to the fact that the count vectors can grow unbounded). We now show how we can reduce this infinite state space to a finite state space. This allows us to compute an $\epsilon$-optimal value function over the resulting finite-dimensionnal belief space using standard POMDP techniques. Various methods for belief tracking in the infinite model are also presented.

## 4.1 Approximate Finite Model

We first present an upper bound on the value difference between two states that differ only by their model estimate $\phi$ and $\psi$. This bound uses the following definitions: given $\phi, \phi' \in \mathcal{T}$, and $\psi, \psi' \in \mathcal{O}$, define $D^{sa}_S(\phi, \phi') = \sum_{s' \in S} \left| T^{sas'}_\phi - T^{sas'}_{\phi'} \right|$ and $D^{sa}_Z(\psi, \psi') = \sum_{z \in Z} \left| O^{saz}_\psi - O^{saz}_{\psi'} \right|$, and $\mathcal{N}^{sa}_\phi = \sum_{s' \in S} \phi^a_{ss'}$ and $\mathcal{N}^{sa}_\psi = \sum_{z \in Z} \psi^a_{sz}$.

**Theorem 4.1.** *Given any $\phi, \phi' \in \mathcal{T}$, $\psi, \psi' \in \mathcal{O}$, and $\gamma \in (0, 1)$, then for all $t$:*

$$
\sup_{\alpha_t \in \Gamma_t, s \in S} |\alpha_t(s, \phi, \psi) - \alpha_t(s, \phi', \psi')| \leq \frac{2\gamma ||R||_\infty}{(1-\gamma)^2} \sup_{s, s' \in S, a \in A} \left[ D^{sa}_S(\phi, \phi') + D^{s'a}_Z(\psi, \psi') \right.
$$

$$
\left. + \frac{4}{\ln(\gamma^{-e})} \left( \frac{\sum_{s'' \in S} |\phi^a_{ss''} - \phi'^a_{ss''}|}{(\mathcal{N}^{sa}_\phi + 1)(\mathcal{N}^{sa}_{\phi'} + 1)} + \frac{\sum_{z \in Z} |\psi^a_{s'z} - \psi'^a_{s'z}|}{(\mathcal{N}^{s'a}_\psi + 1)(\mathcal{N}^{s'a}_{\psi'} + 1)} \right) \right]
$$

*Proof.* Proof available in [11] finds a bound on a 1-step backup and solves the recurrence. □

We now use this bound on the $\alpha$-vector values to approximate the space of Dirichlet parameters within a finite subspace. We use the following definitions: given any $\epsilon > 0$, define $\epsilon' = \frac{\epsilon(1-\gamma)^2}{8\gamma ||R||_\infty}$, $\epsilon'' = \frac{\epsilon(1-\gamma)^2 \ln(\gamma^{-e})}{32\gamma ||R||_\infty}$, $N^\epsilon_S = \max \left( \frac{|S|(1+\epsilon')}{\epsilon'}, \frac{1}{\epsilon''} - 1 \right)$ and $N^\epsilon_Z = \max \left( \frac{|Z|(1+\epsilon')}{\epsilon'}, \frac{1}{\epsilon''} - 1 \right)$.

**Theorem 4.2.** *Given any $\epsilon > 0$ and $(s, \phi, \psi) \in S'$ such that $\exists a \in A, s' \in S, \mathcal{N}^{s'a}_\phi > N^\epsilon_S$ or $\mathcal{N}^{s'a}_\psi > N^\epsilon_Z$, then $\exists (s, \phi', \psi') \in S'$ such that $\forall a \in A, s' \in S, \mathcal{N}^{s'a}_{\phi'} \leq N^\epsilon_S$ and $\mathcal{N}^{s'a}_{\psi'} \leq N^\epsilon_Z$ where $|\alpha_t(s, \phi, \psi) - \alpha_t(s, \phi', \psi')| < \epsilon$ holds for all $t$ and $\alpha_t \in \Gamma_t$.*

*Proof.* Proof available in [11]. □

Theorem 4.2 suggests that if we want a precision of $\epsilon$ on the value function, we just need to restrict the space of Dirichlet parameters to count vectors $\phi \in \tilde{\mathcal{T}}_\epsilon = \{\phi \in \mathbb{N}^{|S|^2|A|} | \forall a \in A, s \in S, 0 < \mathcal{N}_\phi^{sa} \le N_S^\epsilon\}$ and $\psi \in \tilde{\mathcal{O}}_\epsilon = \{\psi \in \mathbb{N}^{|S||A||Z|} | \forall a \in A, s \in S, 0 < \mathcal{N}_\psi^{sa} \le N_Z^\epsilon\}$. Since $\tilde{\mathcal{T}}_\epsilon$ and $\tilde{\mathcal{O}}_\epsilon$ are finite, we can define a finite approximate BAPOMDP as the tuple $(\tilde{S}_\epsilon, A, Z, \tilde{T}_\epsilon, \tilde{O}_\epsilon, \tilde{R}_\epsilon, \gamma)$ where $\tilde{S}_\epsilon = S \times \tilde{\mathcal{T}}_\epsilon \times \tilde{\mathcal{O}}_\epsilon$ is the finite state space. To define the transition and observation functions over that finite state space, we need to make sure that when the count vectors are incremented, they stay within the finite space. To achieve, this we define a projection operator $\mathcal{P}_\epsilon : S' \to \tilde{S}_\epsilon$ that simply projects every state in $S'$ to their closest state in $\tilde{S}_\epsilon$.

**Definition 4.1.** *Let* $d : S' \times S' \to \mathbb{R}$ *be defined such that:*

$$d(s, \phi, \psi, s', \phi', \psi') = \begin{cases} \frac{2\gamma||R||_\infty}{(1-\gamma)^2} \sup\limits_{s,s' \in S, a \in A} \left[ D_S^{sa}(\phi, \phi') + D_Z^{s'a}(\psi, \psi') \right. \\ \left. + \frac{4}{\ln(\gamma^{-e})} \left( \frac{\sum_{s'' \in S} |\phi_{ss''}^a - \phi_{ss''}'^a|}{(\mathcal{N}_\phi^{as}+1)(\mathcal{N}_{\phi'}^{as}+1)} + \frac{\sum_{z \in Z} |\psi_{s'z}^a - \psi_{s'z}'^a|}{(\mathcal{N}_\psi^{as'}+1)(\mathcal{N}_{\psi'}^{as'}+1)} \right) \right], & \text{if } s = s' \\ \frac{8\gamma||R||_\infty}{(1-\gamma)^2} \left( 1 + \frac{4}{\ln(\gamma^{-e})} \right) + \frac{2||R||_\infty}{(1-\gamma)}, & \text{otherwise.} \end{cases}$$

**Definition 4.2.** *Let* $\mathcal{P}_\epsilon : S' \to \tilde{S}_\epsilon$ *be defined as* $\mathcal{P}_\epsilon(s) = \arg\min\limits_{s' \in \tilde{S}_\epsilon} d(s, s')$

The function $d$ uses the bound defined in Theorem 4.1 as a distance between states that only differs by their $\phi$ and $\psi$ vectors, and uses an upper bound on that value when the states differ. Thus $\mathcal{P}_\epsilon$ always maps states $(s, \phi, \psi) \in S'$ to some state $(s, \phi', \psi') \in \tilde{S}_\epsilon$. Note that if $\sigma \in \tilde{S}_\epsilon$, then $\mathcal{P}_\epsilon(\sigma) = \sigma$. Using $\mathcal{P}_\epsilon$, the transition and observation function are defined as follows:

$$\tilde{T}_\epsilon((s, \phi, \psi), a, (s', \phi', \psi')) = \begin{cases} T_\phi^{sas'} O_\psi^{s'az}, & \text{if } (s', \phi', \psi') = \mathcal{P}_\epsilon(s', \phi + \delta_{ss'}^a, \psi + \delta_{s'z}^a) \\ 0, & \text{otherwise.} \end{cases} \quad (4)$$

$$\tilde{O}_\epsilon((s, \phi, \psi), a, (s', \phi', \psi'), z) = \begin{cases} 1, & \text{if } (s', \phi', \psi') = \mathcal{P}_\epsilon(s', \phi + \delta_{ss'}^a, \psi + \delta_{s'z}^a) \\ 0, & \text{otherwise.} \end{cases} \quad (5)$$

These definitions are the same as the one in the infinite BAPOMDP, except that now we add an extra projection to make sure that the incremented count vectors stays in $\tilde{S}_\epsilon$. Finally, the reward function $\tilde{R}_\epsilon : \tilde{S}_\epsilon \times A \to \mathbb{R}$ is defined as $\tilde{R}_\epsilon((s, \phi, \psi), a) = R(s, a)$.

Theorem 4.3 bounds the value difference between $\alpha$-vectors computed with this finite model and the $\alpha$-vector computed with the original model.

**Theorem 4.3.** *Given any* $\epsilon > 0$, $(s, \phi, \psi) \in S'$ *and* $\alpha_t \in \Gamma_t$ *computed from the infinite BAPOMDP. Let* $\tilde{\alpha}_t$ *be the* $\alpha$-*vector representing the same conditionnal plan as* $\alpha_t$ *but computed with the finite BAPOMDP* $(\tilde{S}_\epsilon, A, Z, \tilde{T}_\epsilon, \tilde{O}_\epsilon, \tilde{R}_\epsilon, \gamma)$, *then* $|\tilde{\alpha}_t(\mathcal{P}_\epsilon(s, \phi, \psi)) - \alpha_t(s, \phi, \psi)| < \frac{\epsilon}{1-\gamma}$.

*Proof.* Proof available in [11]. Solves a recurrence over the 1-step approximation in Thm. 4.2. ☐

Because the state space is now finite, solution methods from the literature on finite POMDPs could theoretically be applied. This includes en particular the equations for $\tau(b, a, z)$ and $V^*(b)$ that were presented in Section 2. In practice however, even though the state space is finite, it will generally be very large for small $\epsilon$, such that it may still be intractable, even for small domains. We therefore favor a faster online solution approach, as described below.

## 4.2 Approximate Belief Monitoring

As shown in Theorem 3.1, the number of states with non-zero probability grows exponentially in the planning horizon, thus exact belief monitoring can quickly become intractable. We now discuss different particle-based approximations that allow polynomial-time belief tracking.

**Monte Carlo sampling**: Monte Carlo sampling algorithms have been widely used for sequential state estimation [12]. Given a prior belief $b$, followed by action $a$ and observation $z$, the new belief $b'$ is obtained by first sampling $K$ states from the distribution $b$, then for each sampled $s$ a new state $s'$ is sampled from $T(s, a, \cdot)$. Finally, the probability $O(s', a, z)$ is added to $b'(s')$ and the belief $b'$ is re-normalized. This will capture at most $K$ states with non-zero probabilities. In the context of

BAPOMDPs, we use a slight variation of this method, where $(s, \phi, \psi)$ are first sampled from $b$, and then a next state $s' \in S$ is sampled from the normalized distribution $T_\phi^{sa\cdot} O_\psi^{\cdot az}$. The probability $1/K$ is added directly to $b'(s', \phi + \delta_{ss'}^a, \psi + \delta_{s'z}^a)$.

**Most Probable**: Alternately, we can do the exact belief update at a given time step, but then only keep the $K$ most probable states in the new belief $b'$ and renormalize $b'$.

**Weighted Distance Minimization**: The two previous methods only try to approximate the distribution $\tau(b, a, z)$. However, in practice, we only care most about the agent's expected reward. Hence, instead of keeping the $K$ most likely states, we can keep $K$ states which best approximate the belief's value. As in the Most Probable method, we do an exact belief update, however in this case we fit the posterior distribution using a greedy $K$-means procedure, where distance is defined as in Definition 4.1, weighted by the probability of the state to remove. See [11] for algorithmic details.

### 4.3 Online planning

While the finite model presented in Section 4.1 can be used to find provably near-optimal policies offline, this will likely be intractable in practice due to the very large state space required to ensure good precision. Instead, we turn to online lookahead search algorithms, which have been proposed for solving standard POMDPs [9]. Our approach simply performs dynamic programming over all the beliefs reachable within some fixed finite planning horizon from the current belief. The action with highest return over that finite horizon is executed and then planning is conducted again on the next belief. To further limit the complexity of the online planning algorithm, we used the approximate belief monitoring methods detailed above. Its overall complexity is in $O((|A||Z|)^D C_b)$ where $D$ is the planning horizon and $C_b$ is the complexity of updating the belief.

## 5 Empirical Results

We begin by evaluating the different belief approximations introduced above. To do so, we use a simple online $d$-step lookahead search, and compare the overall expected return and model accuracy in two different problems: the well-known Tiger [5] and a new domain called Follow. Given $T^{sas'}$ and $O^{s'az}$ the exact probabilities of the (unknown) POMDP, the model accuracy is measured in terms of the weighted sum of L1-distance, denoted $WL1$, between the exact model and the probable models in a belief state $b$:

$$
\begin{aligned}
WL1(b) &= \sum_{(s,\phi,\psi) \in S_b'} b(s, \phi, \psi) L1(\phi, \psi) \\
L1(\phi, \psi) &= \sum_{a \in A} \sum_{s' \in S} \left[ \sum_{s \in S} |T_\phi^{sas'} - T^{sas'}| + \sum_{z \in Z} |O_\psi^{s'az} - O^{s'az}| \right]
\end{aligned}
\tag{6}
$$

### 5.1 Tiger

In the Tiger problem [5], we consider the case where the transition and reward parameters are known, but the observation probabilities are not. Hence, there are four unknown parameters: $O_{Ll}$, $O_{Lr}$, $O_{Rl}$, $O_{Rr}$ ($O_{Lr}$ stands for $\Pr(z = hear\_right|s = tiger\_Left, a = Listen)$). We define the observation count vector $\psi = (\psi_{Ll}, \psi_{Lr}, \psi_{Rl}, \psi_{Rr})$. We consider a prior of $\psi_0 = (5, 3, 3, 5)$, which specifies an expected sensor accuracy of $62.5\%$ (instead of the correct $85\%$) in both states. Each simulation consists of 100 episodes. Episodes terminate when the agent opens a door, at which point the POMDP state (i.e. tiger's position) is reset, but the distribution over count vector is carried over to the next episode.

Figures 1 and 2 show how the average return and model accuracy evolve over the 100 episodes (results are averaged over 1000 simulations), using an online 3-step lookahead search with varying belief approximations and parameters. Returns obtained by planning directly with the prior and exact model (without learning) are shown for comparison. Model accuracy is measured on the initial belief of each episode. Figure 3 compares the average planning time per action taken by each approach. We observe from these figures that the results for the Most Probable and Weighted Distance approximations are very similar and perform well even with few particles (lines are overlapping in many places, making Weighted Distance results hard to see). On the other hand, the performance of Monte Carlo is significantly affected by the number of particles and had to use much more par-

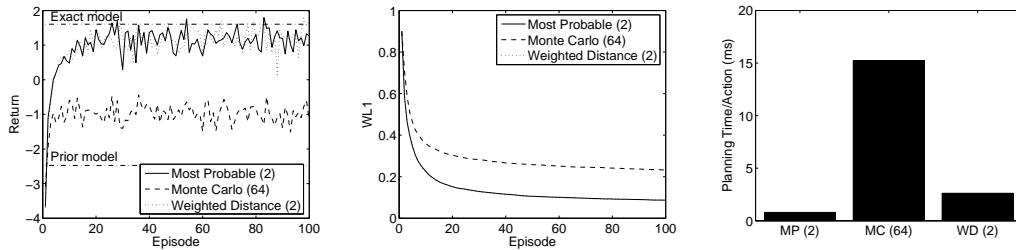

Figure 1: Return with different belief approximations.

Figure 2: Model accuracy with different belief approximations.

Figure 3: Planning Time with different belief approximations.

ticles (64) to obtain an improvement over the prior. This may be due to the sampling error that is introduced when using fewer samples.

## 5.2 Follow

We propose a new POMDP domain, called Follow, inspired by an interactive human-robot task. It is often the case that such domains are particularly subject to parameter uncertainty (due to the difficulty in modelling human behavior), thus this environment motivates the utility of Bayes-Adaptive POMDP in a very practical way. The goal of the Follow task is for a robot to continuously follow one of two individuals in a 2D open area. The two subjects have different motion behavior, requiring the robot to use a different policy for each. At every episode, the target person is selected randomly with $Pr = 0.5$ (and the other is not present). The person's identity is not observable (except through their motion). The state space has two features: a binary variable indicating which person is being followed, and a position variable indicating the person's position relative to the robot ($5 \times 5$ square grid with the robot always at the center). Initially, the robot and person are at the same position. Both the robot and the person can perform five motion actions $\{NoAction, North, East, South, West\}$. The person follows a fixed stochastic policy (stationary over space and time), but the parameters of this behavior are unknown. The robot perceives observations indicating the person's position relative to the robot: $\{Same, North, East, South, West, Unseen\}$. The robot perceives the correct observation $Pr = 0.8$ and $Unseen$ with $Pr = 0.2$. The reward $R = +1$ if the robot and person are at the same position (central grid cell), $R = 0$ if the person is one cell away from the robot, and $R = -1$ if the person is two cells away. The task terminates if the person reaches a distance of 3 cells away from the robot, also causing a reward of -20. We use a discount factor of 0.9.

When formulating the BAPOMDP, the robot's motion model (deterministic), the observation probabilities and the rewards are assumed to be known. We maintain a separate count vector for each person, representing the number of times they move in each direction, i.e. $\phi^1 = (\phi^1_{NA}, \phi^1_N, \phi^1_E, \phi^1_S, \phi^1_W)$, $\phi^2 = (\phi^2_{NA}, \phi^2_N, \phi^2_E, \phi^2_S, \phi^2_W)$. We assume a prior $\phi^1_0 = (2, 3, 1, 2, 2)$ for person 1 and $\phi^2_0 = (2, 1, 3, 2, 2)$ for person 2, while in reality person 1 moves with probabilities $Pr = (0.3, 0.4, 0.2, 0.05, 0.05)$ and person 2 with $Pr = (0.1, 0.05, 0.8, 0.03, 0.02)$. We run 200 simulations, each consisting of 100 episodes (of at most 10 time steps). The count vectors' distributions are reset after every simulation, and the target person is reset after every episode. We use a 2-step lookahead search for planning in the BAPOMDP.

Figures 4 and 5 show how the average return and model accuracy evolve over the 100 episodes (averaged over the 200 simulations) with different belief approximations. Figure 6 compares the planning time taken by each approach. We observe from these figures that the results for the Weighted Distance approximations are much better both in terms of return and model accuracy, even with fewer particles (16). Monte Carlo fails at providing any improvement over the prior model, which indicates it would require much more particles. Running Weighted Distance with 16 particles require less time than both Monte Carlo and Most Probable with 64 particles, showing that it can be more time efficient for the performance it provides in complex environment.

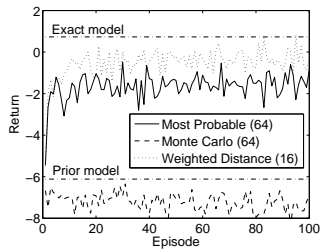
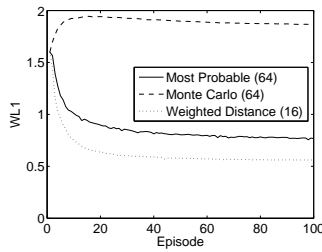
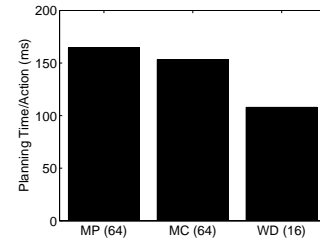

Figure 4: Return with different belief approximations.

Figure 5: Model accuracy with different belief approximations.

Figure 6: Planning Time with different belief approximations.

## 6 Conclusion

The objective of this paper was to propose a preliminary decision-theoretic framework for learning and acting in POMDPs under parameter uncertainty. This raises a number of interesting challenges, including (1) defining the appropriate model for POMDP parameter uncertainty, (2) approximating this model while maintaining performance guarantees, (3) performing tractable belief updating, and (4) planning action sequences which optimally trade-off exploration and exploitation.

We proposed a new model, the Bayes-Adaptive POMDP, and showed that it can be approximated to $\epsilon$-precision by a finite POMDP. We provided practical approaches for belief tracking and online planning in this model, and validated these using two experimental domains. Results in the Follow problem, showed that our approach is able to learn the motion patterns of two (simulated) individuals. This suggests interesting applications in human-robot interaction, where it is often essential that we be able to reason and plan under parameter uncertainty.

## Acknowledgments

This research was supported by the Natural Sciences and Engineering Research Council of Canada (NSERC) and the Fonds Québécois de la Recherche sur la Nature et les Technologies (FQRNT).

## References

[1] R. Dearden, N. Friedman, and N. Andre. Model based bayesian exploration. In *UAI*, 1999.

[2] M. Duff. *Optimal Learning: Computational Procedure for Bayes-Adaptive Markov Decision Processes*. PhD thesis, University of Massachusetts, Amherst, USA, 2002.

[3] P. Poupart, N. Vlassis, J. Hoey, and K. Regan. An analytic solution to discrete bayesian reinforcement learning. In *Proc. ICML*, 2006.

[4] R. Jaulmes, J. Pineau, and D. Precup. Active learning in partially observable markov decision processes. In *ECML*, 2005.

[5] L. P. Kaelbling, M. L. Littman, and A. R. Cassandra. Planning and acting in partially observable stochastic domains. *Artificial Intelligence*, 101:99–134, 1998.

[6] J. Pineau, G. Gordon, and S. Thrun. Point-based value iteration: an anytime algorithm for POMDPs. In *IJCAI*, pages 1025–1032, Acapulco, Mexico, 2003.

[7] M. Spaan and N. Vlassis. Perseus: randomized point-based value iteration for POMDPs. *JAIR*, 24:195–220, 2005.

[8] T. Smith and R. Simmons. Heuristic search value iteration for POMDPs. In *UAI*, Banff, Canada, 2004.

[9] S. Paquet, L. Tobin, and B. Chaib-draa. An online POMDP algorithm for complex multiagent environments. In *AAMAS*, 2005.

[10] Jonathan Baxter and Peter L. Bartlett. Infinite-horizon policy-gradient estimation. *Journal of Artificial Intelligence Research (JAIR)*, 15:319–350, 2001.

[11] Stéphane Ross, Brahim Chaib-draa, and Joelle Pineau. Bayes-adaptive pomdps. Technical Report SOCS-TR-2007.6, McGill University, 2007.

[12] A. Doucet, N. de Freitas, and N. Gordon. *Sequential Monte Carlo Methods In Practice*. Springer, 2001.
